# Learning Mackey-Glass from 25 examples, Plus or Minus 2

Mark Plutowski* Garrison Cottrell* Halbert White**
Institute for Neural Computation
*Department of Computer Science and Engineering
**Department of Economics
University of California, San Diego
La Jolla, CA 92093

## Abstract

We apply active exemplar selection (Plutowski & White, 1991; 1993) to predicting a chaotic time series. Given a fixed set of examples, the method chooses a concise subset for training. Fitting these exemplars results in the entire set being fit as well as desired. The algorithm incorporates a method for regulating network complexity, automatically adding exemplars and hidden units as needed. Fitting examples generated from the Mackey-Glass equation with fractal dimension 2.1 to an rmse of 0.01 required about 25 exemplars and 3 to 6 hidden units. The method requires an order of magnitude fewer floating point operations than training on the entire set of examples, is significantly cheaper than two contending exemplar selection techniques, and suggests a simpler active selection technique that performs comparably.

## 1 Introduction

Plutowski & White (1991; 1993), have developed a method of *active selection* of training exemplars for network learning. Active selection uses information about the state of the network when choosing new exemplars. The approach uses the statistical sampling criterion Integrated Squared Bias (ISB) to derive a greedy selection method that picks the training example maximizing the decrement in this measure. (ISB is a special case of the more familiar Integrated Mean Squared Error in the case that noise variance is zero.) We refer to this method as $\Delta ISB$. The method automatically regulates network complexity by growing the network as necessary

to fit the selected exemplars, and terminates when the model fits the entire set of available examples to the desired accuracy. Hence the method is a nonparametric regression technique. In this paper we show that the method is practical by applying it to the Mackey-Glass time series prediction task. We compare $\Delta ISB$ with the method of training on all the examples. $\Delta ISB$ consistently learns the time series from a small subset of the available examples, finding solutions equivalent to solutions obtained using all of the examples. The networks obtained by $\Delta ISB$ consistently perform better on test data for single step prediction, and do at least as well at iterated prediction, but are trained at much lower cost.

Having demonstrated that this particular type of exemplar selection is worthwhile, we compare $\Delta ISB$ with three other exemplar selection methods which are easier to code and cost less to compute. We compare the total cost of training, as well as the size of the exemplar sets selected. One of the three contending methods was suggested by the $\Delta ISB$ algorithm, and is also an active selection technique, as its calculation involves the network state. Among the four exemplar selection methods, we find that the two active selection methods provide the greatest computational savings and select the most concise training sets.

## 2   The Method

We are provided with a set of $N$ "candidate" examples of the form $(x_i, g(x_i))$. Given $g$, we can denote this as $x^N$. Let $f(\cdot, w)$ denote the network function parameterized by weights $w$. For a particular subset of the examples denoted $x^n$, let $w_n = w_n(x^n)$ minimize

$$\frac{1}{n} \sum_{i=1}^{n} (g(x_i) - f(x_i, w))^2.$$

Let $w^*$ be the "best" set of weights, which minimizes

$$\int (g(x) - f(x, w^*))^2 \mu(dx),$$

where $\mu$ is the distribution over the inputs. Our objective is to select a subset $x^n$ of $x^N$ such that $n \ll N$, while minimizing $\int (f(x, w_n) - f(x, w^*))^2 \mu(dx)$. Thus, we desire a subset representative of the whole set. We choose the $x^n \subset x^N$ giving weights $w_n$ that minimize the Integrated Squared Bias (ISB):

$$ISB(x^n) = \int (g(x) - f(x, w_n))^2 \mu(dx). \tag{1}$$

We generate $x^n$ incrementally. Given a candidate example $\breve{x}_{n+1}$, let $\breve{x}^{n+1} = (x^n, \breve{x}_{n+1})$. Selecting $x^1$ optimally with respect to (1) is straightforward. Then given $x^n$ minimizing $ISB(x^n)$, we opt to select $x_{n+1} \in x^N$ maximizing $ISB(x^n) - ISB(\breve{x}^{n+1})$. Note that using this property for $x_{n+1}$ will not necessarily deliver the globally optimal solution. Nevertheless, this approach permits a computationally feasible and attractive method for sequential selection of training examples.

Choosing $x_{n+1}$ to maximize this decrement directly is expensive. We use the following simple approximation (see Plutowski & White, 1991) for justification): Given $x^n$, select $x_{n+1} \in \arg\max_{\check{x}_{n+1}} \Delta ISB(\check{x}_{n+1}|x^n)$, where

$$\Delta ISB(\check{x}_{n+1}|x^n) = \Delta \check{w}_{n+1}' \sum_{i=1}^{N} \nabla_w f(x_i, w_n)(g(x_i) - f(x_i, w_n)),$$

$$\Delta \check{w}_{n+1}' = (g(\check{x}_{n+1}) - f(\check{x}_{n+1}, w_n))' \nabla_w f(\check{x}_{n+1}, w_n)'[H(x^n, w_n)]^{-1},$$

and

$$H(x^n, w_n) = \sum_{i=1}^{n} \nabla_w f(x_i, w_n) \nabla_w f(x_i, w_n)'.$$

In practice we approximate $H$ appropriately for the task at hand. Although we arrive at this criterion by making use of approximations valid for large $n$, this criterion has an appealing interpretation as picking the single example having individual error gradient most highly correlated with the average error gradient of the entire set of examples. Learning with this example is therefore likely to be especially informative. The $\Delta ISB$ criterion thus possesses heuristic appeal in training sets of any size.

## 3   The Algorithm

Before presenting the algorithm we first explain certain implementation details. We integrated the $\Delta ISB$ criterion with a straightforward method for regulating network complexity. We begin with a small network and an initial training set composed of a single exemplar. When a new exemplar is added, if training stalls, we randomize the network weights and restart training. After 5 stalls, we grow the network by adding another unit to each hidden layer.

Before we can select a new exemplar, we require that the network fit the current training set "sufficiently well." Let $e_n(\tilde{x}^m)$ measure the rmse (root mean squared error) network fit over $m$ arbitrary examples $\tilde{x}^m$ when trained on $x^n$. Let $F_n \in \Re^+$ denote the rmse fit we require over the current set of $n$ exemplars before selecting a new one. Let $F_N \in \Re^+$ denote the rmse fit desired over all $N$ examples. (Our goal is $e_n(x^N) \leq F_N$.) It typically suffices to set $F_n = F_N$, that is, to train to a fit over the exemplars which is at least as stringent as the fit desired over the entire set (normalized for the number of exemplars.) However, active selection sometimes chooses a new exemplar "too close" to previously selected exemplars even when this is the case. This is easy to detect, and in this case we reject the new exemplar and continue with training.

We use an "exemplar spacing" parameter $d$ to detect when a new exemplar is too close to a previous selection. Two examples $x_i$ and $x_j$ are "close" in this sense if they are within Euclidean distance $d$, and if additionally $|g(x_i) - g(x_j)| \leq F_N$. The additional condition allows the new exemplar to be accepted even when it is close to a previous selection in input space, provided it is sufficiently far away in the output space. In our experiments, the input and output space are of the same scale, so we set $d = F_N$. When a new selection is too close to a current exemplar, we reject the

new selection, reduce $F_n$ by 20%, and continue training, resetting $F_n = F_N$ when a subsequent selection is appended to the current training set. We now outline the algorithm:

**Initialize:**

- Specify user-set parameters: initial network size, the desired fit $F_N$, the exemplar spacing parameter, and the maximum number of restarts.
- Select the first training set, $x^1 = \{x_1\}$. Set $n = 1$ and $F_n = F_N$. Train the network on $x^1$ until $e_n(x^1) \le F_n$.

**While**$(e_n(x^N) > F_N)$ {
    Select a new exemplar, $x_{n+1} \in x^N$, maximizing $\Delta ISB$.
    **If** $(x_{n+1}$ is "too close" to any $x \in x^n)$ {
        Reject $x_{n+1}$
        Reduce $F_n$ by 20%. }
    **Else** {
        Append $x_{n+1}$ to $x^n$.
        Increment $n$.
        Set $F_n = F_N$. }
    **While**$(e_n(x^n) > F_n)$ {
        Train the network on the current training set $x^n$,
        restarting and growing as necessary. }}

## 4   The Problem

We generated the data from the Mackey-Glass equation (Mackey & Glass, 1977), with $\tau = 17$, $a = 0.2$, and $b = 0.1$. We integrated the equation using fourth order Runge-Kutta with step size 0.1, and the history initialized to 0.5. We generated two data sets. We iterated the equation for 100 time steps before beginning sampling; this marks $t = 0$. The next 1000 time steps comprise Data Set 1. We generated Data Set 2 from the 2000 examples following $t = 5000$.

We used the standard feed-forward network architecture with $[0, 1]$ sigmoids and one or two hidden layers. Denoting the time series as $x(t)$, the inputs were $x(t), x(t - 6), x(t - 12), x(t - 18)$, and the desired output is $x(t + 6)$ (Lapedes & Farber, 1987). We used conjugate gradient optimization for all of the training runs. The line search routine typically required 5 to 7 passes through the data set for each downhill step, and was restricted to use no more than 10.

Initially, the single hidden layer network has a single hidden unit, and the 2 hidden layer network has 2 units per hidden layer. A unit is added to each hidden layer when growing either architecture. All methods use the same growing procedure. Thus, other exemplar selection techniques are implemented by modifying how the next training set is obtained at the beginning of the outer while loop. The method of using all the training examples uses only the inner while loop.

In preliminary experiments we evaluated sensitivity of $\Delta ISB$ to the calculation of $H$. We compared two ways of estimating $H$, in terms of the number of exemplars

selected and the total cost of training. The first approach uses the diagonal terms of $H$ (Plutowski & White, 1993). The second approach replaces $H$ with the identity matrix. Evaluated over 10 separate runs, fitting 500 examples to an rmse of 0.01, $\Delta ISB$ gave similar results for both approaches, in terms of total computation used and the number of exemplars selected. Here, we used the second approach.

## 5  The Comparisons

We performed a number of experiments, each comparing the $\Delta ISB$ algorithm with competing training methods. The competing methods include the conventional method of using all the examples, henceforth referred to as "the strawman," as well as three other data selection techniques. In each comparison we denote the *cost* as the total number of floating point multiplies (the number of adds and divides is always proportional to this count).

For each comparison we ran two sets of experiments. The first compares the total cost of the competing methods as the fit requirement is varied between 0.02, 0.015, and 0.01, using the first 500 examples from Data Set 1. The second compares the cost as the size of the "candidate" set (the set of available examples) is varied using the first 500, 625, 750, 875, and 1000 examples of Data Set 1, and a tolerance of 0.01. To ensure that each method is achieving a comparable fit over novel data, we evaluated each network over a test set. The generalization tests also looked at the iterated prediction error (IPE) over the candidate set and test set (Lapedes & Farber, 1987). Here we start the network on the first example from the set, and feed the output back into the network to obtain predictions in multiples of 6 time steps. Finally, for each of these we compare the final network sizes. Each data point reported is an average of five runs. For brevity, we only report results from the two hidden layer networks.

## 6  Comparison With Using All the Examples

We first compare $\Delta ISB$ with the conventional method of using all the available examples, which we will refer to as "the strawman." For this test, we used the first 500 examples of Data Set 1. For the two hidden layer architecture, each method required 2 units per hidden layer for a fit of 0.02 and 0.015 rmse, and from 3 to 4 (typically 3) units per hidden layer for a fit of 0.01 rmse. While both methods did quite well on the generalization tests, $\Delta ISB$ clearly did better. Whereas the strawman networks do slightly worse on the test set than on the candidate set, networks trained by $\Delta ISB$ tended to give test set fits close to the desired (training) fit. This is partially due to the control flow of the algorithm, which often fits the candidate set better than necessary. However, we also observed $\Delta ISB$ networks exhibited a test set fit better than the candidate set fit 7 times over these 15 training runs. This never occurred over any of the strawman runs.

Overall, $\Delta ISB$ networks performed at least as well as the strawman with respect to IPE. Figure 1a shows the second half of Data Set 1, which is novel to this network, plotted along with the iterated prediction of a $\Delta ISB$ network to a fit of 0.01, giving an IPE of 0.081 rmse, the median IPE observed for this set of five runs. Figure 1b shows the iterated prediction over the first 500 time steps of Data Set 2, which is

4500 time steps later than the training set. The IPE is 0.086 rmse, only slightly worse than over the "nearer" test set. This fit required 22 exemplars. Generalization tests were excellent for both methods, although $\Delta ISB$ was again better overall. $\Delta ISB$ networks performed better on Data Set 2 than they did on the candidate set 9 times out of the 25 runs; this never occurred for the strawman. These effects demand closer study before using them to infer that data selection can introduce a beneficial bias. However, they do indicate that the $\Delta ISB$ networks performed at least as well as the strawman, ensuring the validity of our cost comparisons.

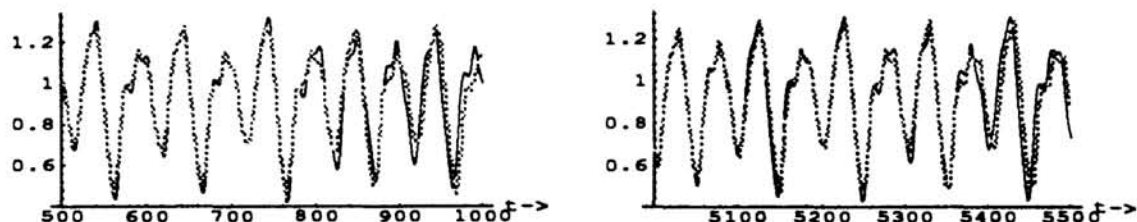

Figure 1: Iterated prediction for a 2 hidden layer network trained to 0.01 rmse over the first 500 time steps of Data Set 1. The dotted line gives the network prediction; the solid line is the target time series. Figure 1a, on the left, is over the next (consecutive) 500 time steps of Data Set 1, with IPE = 0.081 rmse. Figure 1b, on the right, is over the first 500 steps of Data Set 2, with IPE = 0.086 rmse. This network was typical, being the median IPE of 5 runs.

Figure 2a shows the average total cost versus required fit $F_N$ for each method. The strawman required 109, 115, and 4740 million multiplies for the respective tolerances, whereas $\Delta ISB$ required 8, 28, and 219 million multiplies, respectively. The strawman is severely penalized by a tighter fit because growing the network to fit requires expensive restarts using all of the examples. Figure 2b shows the average total cost versus the candidate set sizes. One reason for the difference is that $\Delta ISB$ tended to select smaller networks. For candidate sets of size 500, 625, 750 and 875, each method typically required 3 units per hidden layer, occasionally 4. Given 1000 examples, the strawman selected networks larger than 3 hidden units per layer over twice as often as $\Delta ISB$. $\Delta ISB$ also never required more than 4 hidden units per layer, while the strawman sometimes required 6. This suggests that the growing technique is more likely to fit the data with a smaller network when exemplar selection is used.

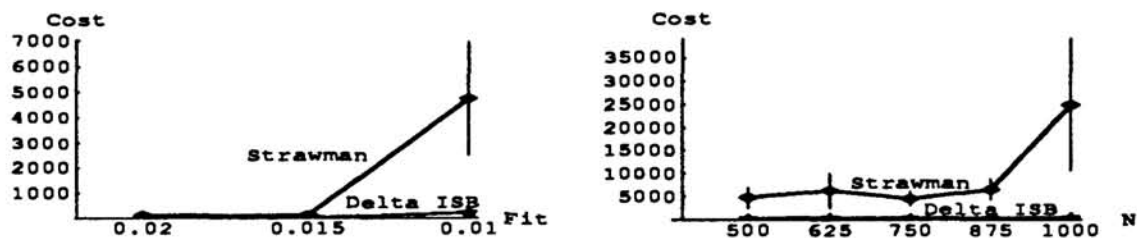

Figure 2: Cost (in millions of multiplies) of training $\Delta ISB$, compared to the Strawman. Figure 2a on the left gives total cost versus the desired fit, and Figure 2b on the right gives total cost versus the number of candidate examples. Each point is the average of 5 runs; the error bars are equal in width to twice the standard deviation.

# 7  Contending Data Selection Techniques

The results above clearly demonstrate that exemplar selection can cut the cost of training dramatically. In what follows we compare $\Delta ISB$ with three other exemplar selection techniques. Each of these is easier to code and cheaper to compute, and are considerably more challenging contenders than the strawman. In addition to comparing the overall training cost we will also evaluate their data compression ability by comparing the size of the exemplar sets each one selects. We proceed in the same manner as with $\Delta ISB$, sequentially growing the training set as necessary, until the candidate set fit is as desired.

Two of these contending techniques do not depend upon the state of the network, and are therefore are not "Active Selection" methods. *Random Selection* selects an example randomly from the candidate set, without replacement, and appends it to the current exemplar set. *Uniform Grid* exploits the time series representation of our data set to select training sets composed of exemplars evenly spaced at regular intervals in time. Note that *Uniform Grid* does not append a single exemplar to the training set, rather it selects an entirely new set of exemplars each time the training set is grown. Note further that this technique relies heavily upon the time series representation. The problem of selecting exemplars uniformly spaced in the 4 dimensional input space would be much more difficult to compute.

The third method, "*Maximum Error*," was suggested by the $\Delta ISB$ algorithm, and is also an Active Selection technique, since it uses the network in selecting new exemplars. Note that the error between the network and the desired value is a component of the $\Delta ISB$ criterion. $\Delta ISB$ need not select an exemplar for which network error is maximum, due to the presence of terms involving the gradient of the network function. In comparison, the *Maximum Error* method selects an exemplar maximizing network error, ignoring gradient information entirely. It is cheaper to compute, typically requiring an order of magnitude fewer multiplies in overhead cost as compared to $\Delta ISB$. This comparison will test, for this particular learning task, whether the gradient information is worth its additional overhead.

## 7.1  Comparison with Random Selection

*Random Selection* fared the worst among the four contenders. However, it still performed better overall than the strawman method. This is probably because the cost due to growing is cheaper, since early on restarts are performed over small training sets. As the network fit improves, the likelihood of randomly selecting an informative exemplar decreases, and *Random Selection* typically reaches a point where it adds exemplars in rapid succession, often doubling the size of the exemplar set in order to attain a slightly better fit. *Random Selection* also had a very high variance in cost and number of exemplars selected.

## 7.2  Comparison with Uniform Grid and Maximum Error

*Uniform Grid* and *Maximum Error* are comparable with $\Delta ISB$ in cost as well as in the size of the selected exemplar sets. Overall, $\Delta ISB$ and *Maximum Error* performed about the same, with *Uniform Grid* finishing respectably in third place. *Maximum Error* was comparable to $\Delta ISB$ in generalization also, doing better on the test set than on the candidate set 10 times out of 40, whereas $\Delta ISB$ did so a

total of 16 times. This occurred only 3 times out of 40 for *Uniform Grid*.

Figure 3a shows that *Uniform Grid* requires more exemplars at all three tolerances, whereas $\Delta ISB$ and *Maximum Error* select about the same number. Figure 3b shows that *Uniform Grid* typically requires about twice as many exemplars as the other two. *Maximum Error* and $\Delta ISB$ selected about the same number of exemplars, typically selecting about 25 exemplars, plus or minus two.

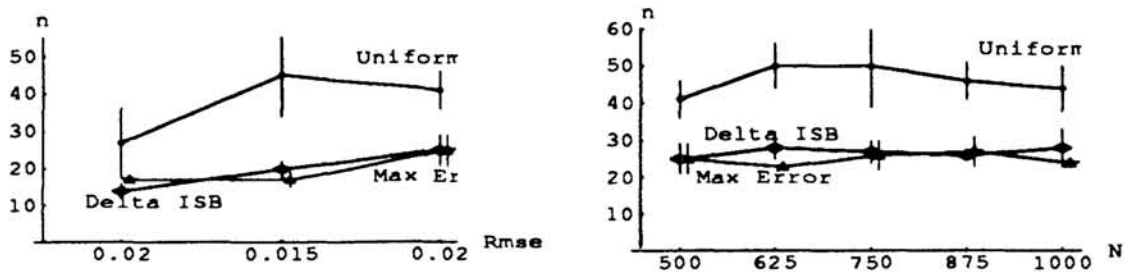

Figure 3:   Number of examples selected by three contending selection techniques: Uniform, $\Delta ISB$ (diamonds) and Max Error (triangles.) Figure 3a on the left gives number of examples selected versus the desired fit, and Figure 3b on the right is versus the number of candidate examples. The two Active Selection techniques selected about 25 exemplars, ±2. Each point is the average of 5 runs; the error bars are equal in width to twice the standard deviation. The datapoints for $\Delta ISB$ and Max Error are shifted slightly in the graph to make them easier to distinguish.

## 8   Conclusions

These results clearly demonstrate that exemplar selection can dramatically lower the cost of training. This particular learning task also showed that Active Selection methods are better overall than two contending exemplar selection techniques.

$\Delta ISB$ and *Maximum Error* consistently selected concise sets of exemplars, reducing the total cost of training despite the overhead associated with exemplar selection. This particular learning task did not provide a clear distinction between the two Active Selection techniques. *Maximum Error* is more attractive on problems of this scope even though we have not justified it analytically, as it performs about as well as $\Delta ISB$ but is easier to code and cheaper to compute.

**Acknowledgements**

This work was supported by NSF grant IRI 92-03532.

**References**

Lapedes, Alan, and Robert Farber. 1987. "Nonlinear signal processing using neural networks. Prediction and system modelling." Los Alamos technical report LA-UR-87-2662.

Mackey, M.C., and L. Glass. 1977. "Oscillation and chaos in physiological control systems." *Science* **197**, 287.

Plutowski, Mark E., and Halbert White. 1991. "Active selection of training examples for network learning in noiseless environments." *Technical Report No. CS91-180,* CSE Dept., UCSD, La Jolla, California.

Plutowski, Mark E., and Halbert White. 1993. "Selecting concise training sets from clean data." *To appear, IEEE Transactions on Neural Networks.* **3**, 1.